# ANALYZING THE ENERGY LANDSCAPES
# OF DISTRIBUTED
# WINNER-TAKE-ALL NETWORKS

David S. Touretzky
School of Computer Science
Carnegie Mellon University
Pittsburgh, PA 15213

## ABSTRACT

DCPS (the Distributed Connectionist Production System) is a neural network with complex dynamical properties. Visualizing the energy landscapes of some of its component modules leads to a better intuitive understanding of the model, and suggests ways in which its dynamics can be controlled in order to improve performance on difficult cases.

## INTRODUCTION

Competition through mutual inhibition appears in a wide variety of network designs. This paper discusses a system with unusually complex competitive dynamics. The system is DCPS, the Distributed Connectionist Production System of Touretzky and Hinton (1988). DCPS is a Boltzmann machine composed of five modules, two of which, labeled "Rule Space" and "Bind Space," are winner-take-all (WTA) networks. These modules interact via their effects on two attentional modules called clause spaces. Clause spaces are another type of competitive architecture based on mutual inhibition, but they do not produce WTA behavior. Both clause spaces provide evidential input to both WTA nets, but since connections are symmetric they also receive top-down "guidance" from the WTA nets. Thus, unlike most other competitive architectures, in DCPS the external input to a WTA net does not remain constant as its state evolves. Rather, the present output of the WTA net helps to determine which evidence will become visible in the clause spaces in the future. This dynamic attentional mechanism allows rule and bind spaces to work together even though they are not directly connected.

DCPS actually uses a distributed version of winner-take-all networks whose operating characteristics differ slightly from the non-distributed version. Analyzing the energy landscapes of DWTA networks has led to a better intuitive understanding of their dynamics. For a complete discussion of the role of DWTA nets in DCPS, and the ways in which insights gained from visualization led to improvements in the system's stochastic search behavior, see [Touretzky, 1989].

## DISTRIBUTED WINNER-TAKE-ALL NETWORKS

In classical WTA nets [Feldman & Ballard, 1982], a unit's output value is a continuous quantity that reflects its activation level. In this paper we analyze a stochastic, distributed version of winner-take-all dynamics using Boltzmann machines, whose units have only binary outputs [Hinton & Sejnowski, 1986]. The amount of evidential input to these units determines its energy gap [Hopfield, 1982], which in turn determines its probability of being active. The network's degree of confidence in a hypothesis is thus reflected in the amount of time the unit spends in the active state. A good instantaneous approximation to strength of support can be obtained by representing each hypothesis with a clique of $k$ independent units looking at a common evidence pool. The number of active units in a clique reflects the strength of that hypothesis. DCPS uses cliques of size 40. Units in rival cliques compete via inhibitory connections

If all units in a clique have identical receptive fields, the result is an "ensemble" Boltzmann machine [Derthick & Tebelskis, 1988]. In DCPS the units have only moderately sized, but highly overlapped, receptive fields, so the amount of evidence individual units perceive is distributed binomially. Small excitatory weights between sibling units help make up for variations in external evidence. They also make states where all the units in a single clique are active be powerful attractors.

Energy tours in a DWTA take one of four basic shapes. Examples may be seen in Figure 1a. Let $e$ be the amount of external evidence available to each unit, $\theta$ the unit's threshold, $k$ the clique size, and $w_s$ the excitatory weight between siblings. The four shapes are:

**Eager vee:** the evidence is above threshold ($e > \theta$). The system is eager to turn units on; energy decreases as the number of active units goes up. We have a broad, deep energy well, which the system will naturally fall into given the chance.

**Reluctant vee:** the evidence is below threshold, but a little bit of sibling influence (fewer than $k/2$ siblings) is enough to make up the difference and put the system over the energy barrier. We have $e < \theta < e + w_s(k-1)/2$. The system is initially reluctant to turn units on because that causes the energy to go up, but once over the hump it willingly turns on more units. With all units in the clique active, the system is in an energy well whose energy is below zero.

**Dimpled peak:** with higher thresholds the total energy of the network may remain above zero even when all units are on. This happens when more than half of the siblings must be active to boost each unit above threshold, i.e., $e + w_s(k-1) > \theta > e + w_s(k-1)/2$. The system can still be trapped in the small energy well that remains, but only at low temperatures. The well is hard to reach since the system must first cross a large energy barrier by traveling far uphill in energy space. Even if it does visit the well, the system may easily bounce out of it again if the well is shallow.

**Smooth peak:** when $\theta > e + w_s(k-1)$, units will be below threshold even with full sibling support. In this case there is no energy well, only a peak. The system wants to turn all units off.

## VISUALIZING ENERGY LANDSCAPES

Let's examine the energy landscape of one WTA space when there is ample evidence in the clause spaces for the winning hypothesis. We select three hypotheses, A, B, and C, with disjoint evidence populations. Let hypothesis B be the best supported one with evidence 100, and let A have evidence 40 and C have evidence 5. We will simplify the situation slightly by assuming that all units in a clique perceive exactly the same evidence. In the left half of Figure 1b we show the energy curves for A, B, and C, using a value of 69 for the unit thresholds.[1] Each curve is generated by starting with all units turned off; units for a particular hypothesis are turned on one at a time until all 40 are on; then they are turned off again one at a time, making the curve symmetric. Since the evidence for hypothesis A is a bit below threshold, its curve is of the "reluctant vee" type. The evidence for hypothesis B is well above threshold, so its curve is an "eager vee." Hypothesis C has almost no evidence; its "dimpled peak" shape is due almost entirely to sibling support. (Sibling weights have a value of $+2$; rival weights a value of $-2$.)

Note that the energy well for B is considerably deeper than for A. This means at moderate temperature the model can pop out of A's energy well, but it is more likely to remain in B's well. The well for B is also somewhat broader than the well for A, making it easier for the B attractor to capture the model; its attractor region spans a larger portion of state space.

The energy tours for hypotheses A, B, and C correspond to traversing three orthogonal edges extending from a corner of a $40 \times 40 \times 40$ cube. A point at location $(x, y, z)$ in this cube corresponds to $x$ A units, $y$ B units, and $z$ C units being active. During the stochastic search, A and B units will be flickering on and off simultaneously, so the model will also visit internal points of the cube not covered in the energy tour diagram. To see these points we will use two additional graphic representations of energy landscapes. First, note that hypothesis C gets so little support that we safely can ignore it and concentrate on A and B. This allows us to focus on just the front face of the state space cube. In Figure 2a, the number of active A units runs from zero to forty along the vertical axis, and the number of active B units runs from zero to forty along the horizontal axis. The arrows at each point on the graph show legal state transitions at zero temperature. For example, at the point where there are are 38 active B units and 3 active A units there are two arrows, pointing down and to the right. This means there are two states the model could enter next: it could either turn off one of the active A units, or turn on one more B unit, respectively. At nonzero temperatures other state transitions

are possible, corresponding to uphill moves in energy space, but these two remain the most probable.

The points in the upper left and lower right corners of Figure 2a are marked by "Y" shapes. These represent point attractors at the bottoms of energy wells; the model will not move out of these states unless the temperature is greater than zero. Other points in state space are said to be within the region of a particular attractor if all legal transition sequences (at $T = 0$) from those points lead eventually to the attractor. The attractor regions of A and B are outlined in the figure. Note that the B attractor covers more area than A, as predicted by its greater breadth in the energy tour diagram. Note also that there is a small ridge between the two attractor regions. From starting points on the ridge the model can end up in either final state.

Figure 2b shows the depths of the two attractors. The energy well for B is substantially deeper than the well for A. Starting at the point in the lower left corner where there are zero A units and zero B units active, the energy falls off immediately when moving in the B direction (right), but rises initially in the A direction (left) before dropping into a modest energy well when most of the A units are on. Points in the interior of the diagram, representing a combination of A and B units active, have higher energies than points along the edges due to the inhibitory connections between units in rival cliques.

We can see from Figures 1b and 2 that the attractor for A, although narrower and shallower than the one for B, is still sizable. This is likely to mislead the model, so that some of the time it will get trapped in the wrong energy well. The fact that there is an attractor for A at all is due largely to sibling support, since the raw evidence for A is less than the rule unit threshold.

We can eliminate the unwanted energy well for A by choosing thresholds that exceed the maximum sibling support of $2 \times 39 = 78$. DCPS uses a value of 119. However, early in the stochastic search the evidence visible in the clause spaces will be lower than at the conclusion of the search; high thresholds combined with low evidence would make the B attractor small and very hard to find. (See the right half of Figure 1c, and Figure 3.) Under these conditions the largest attractor is the one with all units turned off: the null hypothesis. `

## DISCUSSION

Our analysis of energy landscapes pulls us in two directions: we need low thresholds so the correct attractor is broad and easy to find, but we need high thresholds to eliminate unwanted attractors associated with local energy minima. Two solutions have been investigated. The first is to start out with low thresholds and raise them gradually during the stochastic search. This "pulls the rug out from under" poorly-supported hypotheses while giving the model time to find the desired winner. The second solution involves clipping a corner from the state space hypercube so that the model may never have fewer than 40 units active at a time. This prevents the

model from falling into the null attractor. When it attempts to drop the number of active units below 40 it is kicked away from the clipped edge by forcing it to turn on a few inactive units at random.

Although DCPS is a Boltzmann machine it does not search the state space by simulated annealing in the usual sense. True annealing implies a slow reduction in temperature over many update cycles. Stochastic search in DCPS takes place at a single temperature that has been empirically determined to be the model's approximate "melting point." The search is only allowed to take a few cycles; typically it takes less than 10. Therefore the shapes of energy wells and the dynamics of the search are particularly important, as they determine how likely the model is to wander into particular attractor regions.

The work reported here suggests that stochastic search dynamics may be improved by manipulating parameters other than just absolute temperature and cooling rate. Threshold growing and corner clipping appear useful in the case of DWTA nets. Additional details are available in [Touretzky, 1989].

## Acknowledgments

This research was supported by the Office of Naval Research under contract N00014-86-K-0678, and by National Science Foundation grant EET-8716324. I thank Dean Pomerleau, Roni Rosenfeld, Paul Gleichauf, and Lokendra Shastri for helpful comments, and Geoff Hinton for his collaboration in the development of DCPS.

## Footnotes

[1] All the weights and thresholds used in this paper are actual DCPS values taken from [Touretzky & Hinton, 1988].

## References

[1] Derthick, M. A., & Tebelskis, J. M. (1988) "Ensemble" Boltzmann machines have collective computational properties like those of Hopfield and Tank neurons. In D. Z. Anderson (ed.), *Neural Information Processing Systems*. New York: American Institute of Physics.

[2] Feldman, J. A., & Ballard, D. H. (1982) Connectionist models and their properties. *Cognitive Science* 6:205-254.

[3] Hinton, G. E., & Sejnowski, T. J. (1986) Learning and relearning in Boltzmann machines. In D. E. Rumelhart and J. L. McClelland (eds.), *Parallel Distributed Processing: Explorations in the Microstructure of Cognition*, volume 1. Cambridge, MA: Bradford Books/The MIT Press.

[4] Hopfield, J. J. (1982) Neural networks and physical systems with emergent collective computational abilities. *Proceedings of the National Academy of Sciences USA*, **79**:2554-2558.

[5] Touretzky, D. S., & Hinton, G. E. (1988) A distributed connectionist production system. *Cognitive Science* **12**(3):423-466.

[6] Touretzky, D. S. (1989) Controlling search dynamics by manipulating energy landscapes. Technical report CMU-CS-89-113, School of Computer Science, Carnegie Mellon University, Pittsburgh, PA.

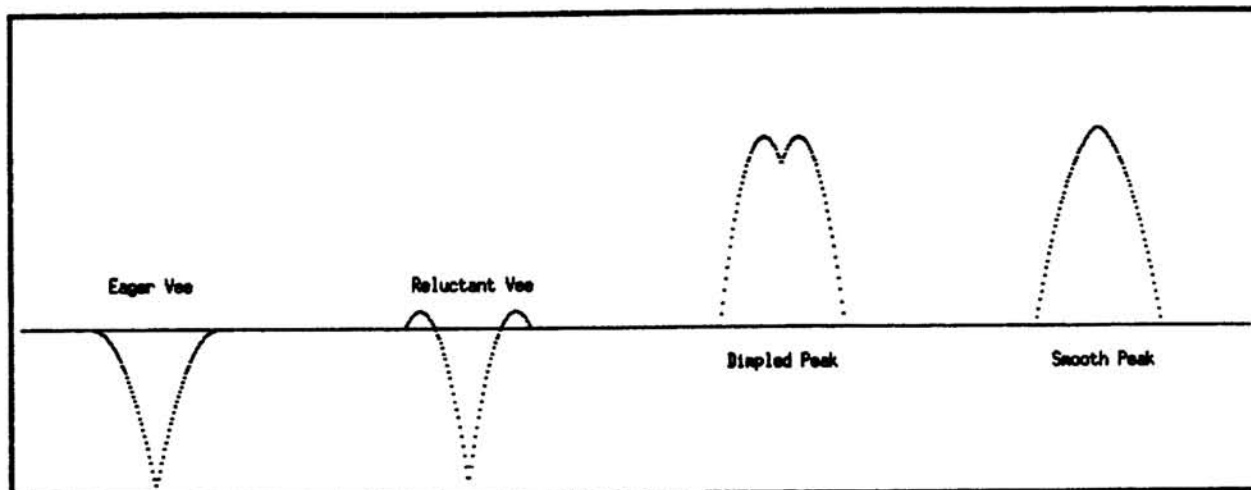

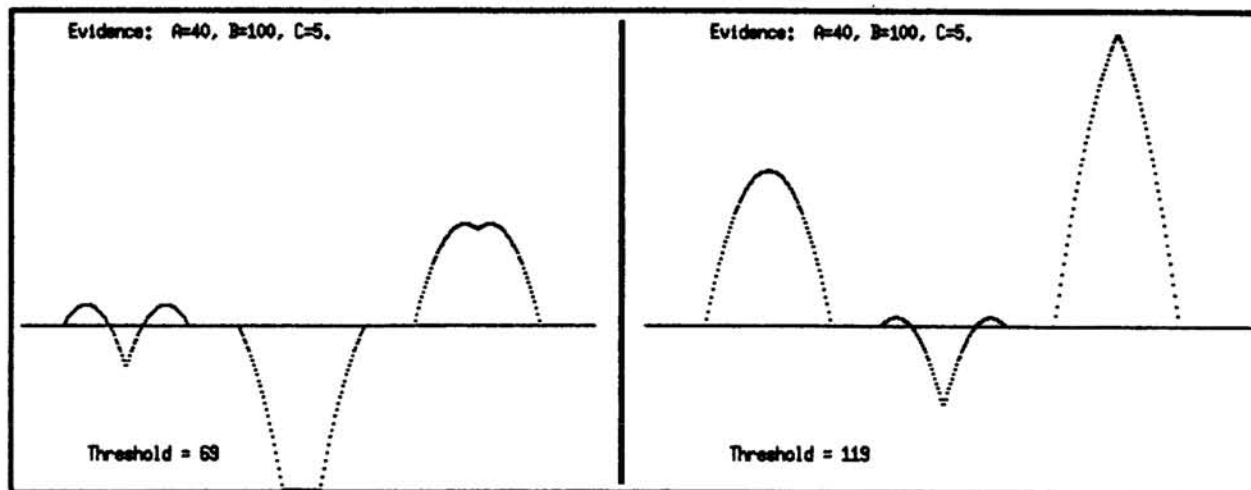

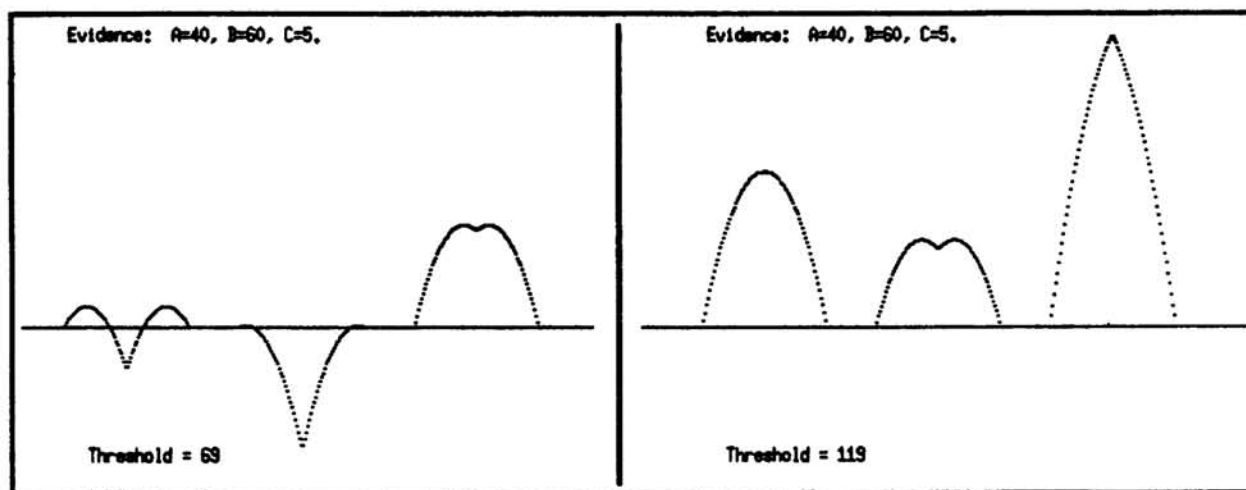

Figure 1: (a) four basic shapes for DWTA energy tours; (b) comparison of low vs. high thresholds in energy tours where there is a high degree of evidence for hypothesis B; (c) corresponding tours with low evidence for B.

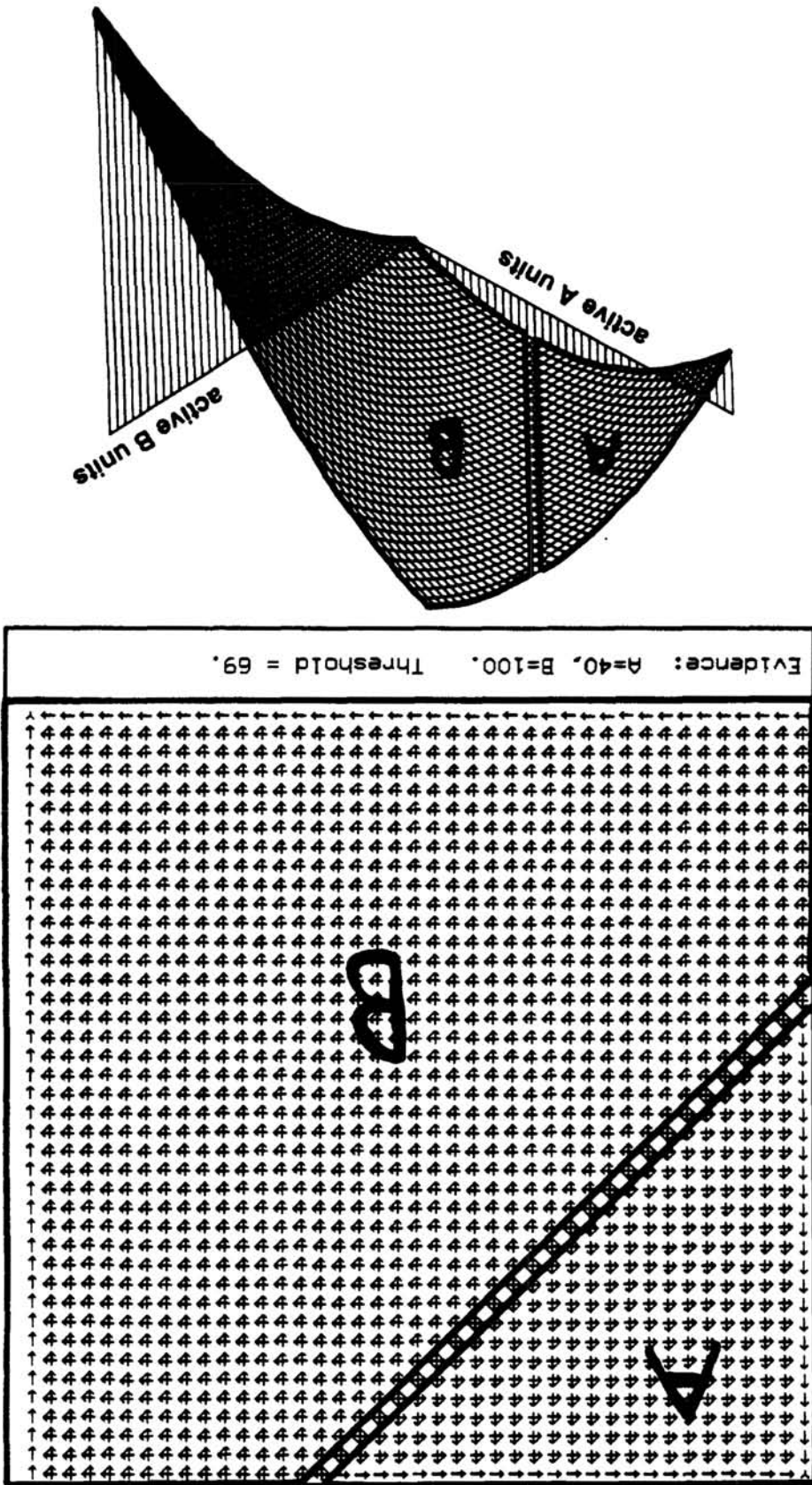

Figure 2: Low thresholds and high evidence, as in the left half of Figure 1b. (a) Legal state transitions at zero temperature; (b) the corresponding energy surface.

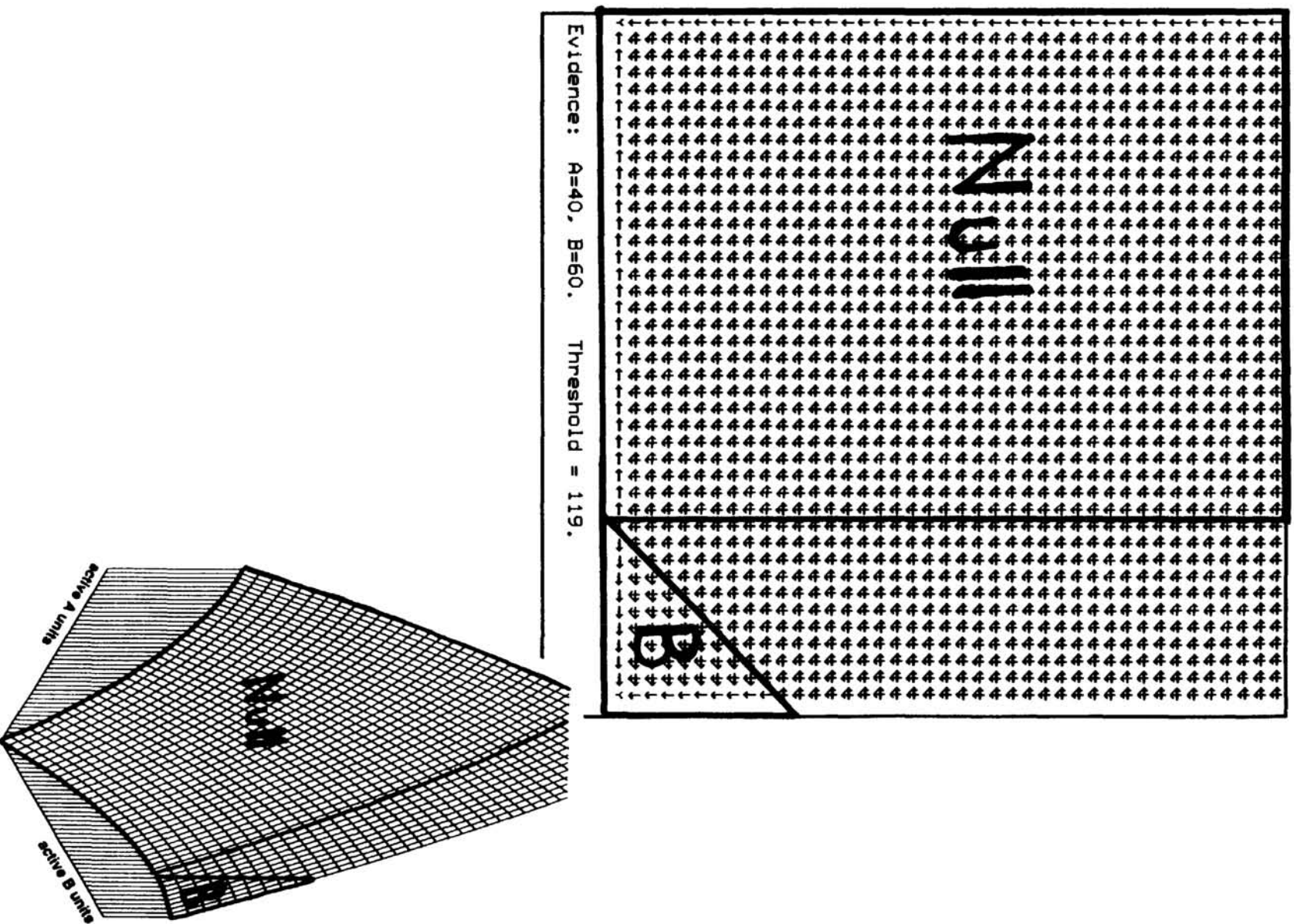

Evidence: A=40, B=60. Threshold = 119.

Figure 3: High thresholds and low evidence, as in the right half of Figure 1c. (a) Legal state transitions at zero temperature; (b) the corresponding energy surface.

